# Semi-supervised Regression via Parallel Field Regularization

**Binbin Lin**  **Chiyuan Zhang**  **Xiaofei He**
State Key Lab of CAD&CG, College of Computer Science, Zhejiang University
Hangzhou 310058, China
{binbinlinzju, chiyuan.zhang.zju, xiaofeihe}@gmail.com

## Abstract

This paper studies the problem of semi-supervised learning from the vector field perspective. Many of the existing work use the graph Laplacian to ensure the smoothness of the prediction function on the data manifold. However, beyond smoothness, it is suggested by recent theoretical work that we should ensure second order smoothness for achieving faster rates of convergence for semi-supervised regression problems. To achieve this goal, we show that the second order smoothness measures the linearity of the function, and the gradient field of a linear function has to be a parallel vector field. Consequently, we propose to find a function which minimizes the empirical error, and simultaneously requires its gradient field to be as parallel as possible. We give a continuous objective function on the manifold and discuss how to discretize it by using random points. The discretized optimization problem turns out to be a sparse linear system which can be solved very efficiently. The experimental results have demonstrated the effectiveness of our proposed approach.

## 1 Introduction

In many machine learning problems, one is often confronted with very high dimensional data. There is a strong intuition that the data may have a lower dimensional intrinsic representation. Various researchers have considered the case when the data is sampled from a submanifold embedded in the ambient Euclidean space. Consequently, learning with the low dimensional manifold structure, or specifically the intrinsic topological and geometrical properties of the data manifold, becomes a crucial problem.

In the past decade, many geometrically motivated approaches have been developed. The early work mainly considers the problem of dimensionality reduction. One hopes that the manifold structure could be preserved in the much lower dimensional Euclidean space. For example, ISOMAP [1] is a global approach which tries to preserve the pairwise geodesic distance on the manifold. Different from ISOMAP, Hessian Eigenmaps (HLLE, [2]) is a local approach for similar purpose. Locally Linear Embedding (LLE, [3]) and Laplacian Eigenmaps (LE, [4]) can be viewed as Laplacian operator based methods which mainly consider the local neighborhood structure of the manifold.

Besides dimensionality reduction, Laplacian based regularization has also been widely employed in semi-supervised learning. These methods construct a nearest neighbor graph over the labeled and unlabeled data to model the underlying manifold structure, and use the graph Laplacian [5] to measure the smoothness of the learned function on the manifold. A variety of semi-supervised learning approaches using the graph Laplacian have been proposed [6, 7, 8]. In semi-supervised regression, some recent theoretical analysis [9] shows that using the graph Laplacian regularizer does not lead to faster minimax rates of convergence. [9] also states that the Laplacian regularizer is way too general for measuring the smoothness of the function. It is further suggested that we

should ensure second order smoothness to achieve faster rates of convergence for semi-supervised regression problems. The Laplacian regularizer is the integral on the norm of the gradient of the function, which is the first order derivative on the function.

In this paper, we design regularization terms that penalize the second order smoothness of the function, i.e., the linearity of the function. Estimating the second order covariant derivative of the function is a very challenging problem. We try to address this problem from vector fields perspective. We show that the gradient field of a linear function has to be a parallel vector field (or parallel field in short). Consequently, we propose a novel approach called Parallel Field Regularization (PFR) to simultaneously find the function and its gradient field, while requiring the gradient field to be as parallel as possible. Specifically, we propose to compute a function and a vector field which satisfy three conditions simultaneously: 1) the function minimizes the empirical error on the labeled data, 2) the vector field is close to the gradient field of the function, 3) the vector field should be as parallel as possible. A novel regularization framework from the vector filed perspective is developed. We give both the continuous and discrete forms of the objective function, and develop an efficient optimization scheme to solve this problem.

## 2 Regularization on the Vector Field

We first briefly introduce semi-supervised learning methods with regularization on the function. Let $\mathcal{M}$ be a $d$-dimensional submanifold in $\mathbb{R}^m$. Given $l$ labeled data points $(x_i, y_i)_{i=1}^l$ on $\mathcal{M}$, we aim to learn a function $f : \mathcal{M} \to \mathbb{R}$ based on the manifold $\mathcal{M}$ and the labeled points $(x_i, y_i)_{i=1}^l$. A framework of semi-supervised learning based on differential operators can be formulated as follows:

$$\arg \min_{f \in C^\infty(\mathcal{M})} E(f) = \frac{1}{l} \sum_{i=1}^l R_0(f(x_i), y_i) + \lambda_1 R_1(f)$$

where $C^\infty(\mathcal{M})$ denotes smooth functions on $\mathcal{M}$, $R_0 : \mathbb{R} \times \mathbb{R} \to \mathbb{R}$ is the loss function and $R_1(f) : C^\infty(\mathcal{M}) \to \mathbb{R}$ is a regularization functional. $R_1$ is often written as a functional norm associated with a differential operator, i.e., $R_1(f) = \int_{\mathcal{M}} \|Df\|^2$ where $D$ is a differential operator. If $D$ is the covariant derivative $\nabla$ on the manifold, then $R_1(f) = \int_{\mathcal{M}} \|\nabla f\|^2 = \int_{\mathcal{M}} f L(f)$ becomes the Laplacian regularizer. If $D$ is the Hessian operator on the manifold, then $R_1(f) = \int_{\mathcal{M}} \|\mathrm{Hess} f\|^2$ becomes the Hessian regularizer.

### 2.1 Parallel Fields and Linear Functions

We first show the relationship between a parallel field and a linear function on the manifold.

**Definition 2.1** (Parallel Field [10]). *A vector field $X$ on manifold $\mathcal{M}$ is a parallel field if*

$$\nabla X \equiv 0,$$

*where $\nabla$ is the covariant derivative on $\mathcal{M}$.*

**Definition 2.2** (Linear Function [10]). *A continuous function $f : \mathcal{M} \to \mathbb{R}$ is said to be linear if*

$$(f \circ \gamma)(t) = f(\gamma(0)) + ct \tag{1}$$

*for each geodesic $\gamma$.*

A function $f$ is linear means that it varies linearly along the geodesics of the manifold. It is a natural extension of linear functions on Euclidean space.

**Proposition 2.1.** [10] *Let $V$ be a parallel field on the manifold. If it is also a gradient field for function $f$, $V = \nabla f$, then $f$ is a linear function on the manifold.*

This proposition tells us the relationship between a parallel field and a linear function on the manifold.

### 2.2 Objective Function

We aim to design regularization terms that penalize the second order smoothness of the function. Following the above analysis, we first approximate gradient field of the prediction function by a

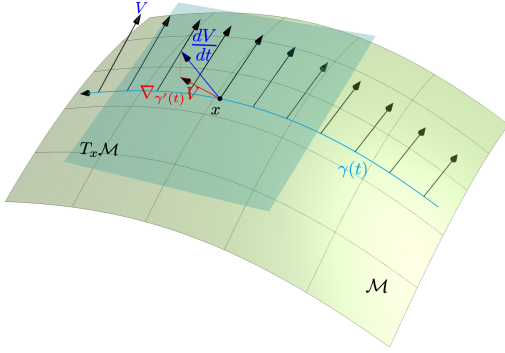

Figure 1: Covariant derivative demonstration. Let $V, Y$ be two vector fields on manifold $\mathcal{M}$. Given a point $x \in \mathcal{M}$, we show how to compute the vector $\nabla_Y V|_x$. Let $\gamma(t)$ be a curve on $\mathcal{M}$: $\gamma : I \to \mathcal{M}$ which satisfies $\gamma(0) = x$ and $\gamma'(0) = Y_x$. Then the covariant derivative along the direction $\frac{d\gamma(t)}{dt}|_{t=0}$ can be computed by projecting $\frac{dV}{dt}|_{t=0}$ to the tangent space $T_x\mathcal{M}$ at $x$. In other words, $\nabla_{\gamma'(0)}V|_x = P_x(\frac{dV}{dt}|_{t=0})$, where $P_x : v \in \mathbb{R}^m \to P_x(v) \in T_x\mathcal{M}$ is the projection matrix. It is not difficult to check that the computation of $\nabla_Y V|_x$ is independent to the choice of the curve $\gamma$.

vector field, then we require the vector field to be as parallel as possible. Therefore, we try to learn the function $f$ and its gradient field $\nabla f$ simultaneously. Formally, we propose to learn a function $f$ and a vector field $V$ on the manifold with two constraints:

- The vector field $V$ should be close to the gradient field $\nabla f$ of $f$, which can be formularized as follows:

$$\min_{f \in C^\infty, V} R_1(f, V) = \int_{\mathcal{M}} \|\nabla f - V\|^2 \tag{2}$$

- The vector field $V$ should be as parallel as possible:

$$\min_V R_2(V) = \int_{\mathcal{M}} \|\nabla V\|_F^2 \tag{3}$$

where $\nabla$ is the covariant derivative on the manifold, $\| \cdot \|_F$ denotes the Frobenius norm.

In the following, we provide some detailed explanation of $R_2(V)$. $\nabla V$ measures the change of the vector field $V$. If $\nabla V$ vanishes, then $V$ is a parallel field. For a given direction $Y_x$ at $x \in \mathcal{M}$, the geometrical meaning of $\nabla_Y V|_x$ is demonstrated in Fig. 1. For a fixed point $x \in \mathcal{M}$, $\nabla V|_x$ is a linear map on the tangent space $T_x\mathcal{M}$. According to the definition of Frobenius norm, we have

$$\|\nabla V\|_F^2 = \sum_{i,j=1}^d (g(\nabla_{\partial_i} V, \partial_j))^2 = \sum_{i=1}^d (g(\nabla_{\partial_i} V, \nabla_{\partial_i} V)) \tag{4}$$

where $g$ is the Riemannian metric on $\mathcal{M}$ and $\partial_1, \dots, \partial_d$ is an orthonormal basis of $T_x\mathcal{M}$.

Naturally, we propose the following objective function based on vector field regularization:

$$\arg\min_{f \in C^\infty(\mathcal{M}), V} E(f, V) = \frac{1}{l} \sum_{i=1}^l R_0(x_i, y_i, f) + \lambda_1 R_1(f, V) + \lambda_2 R_2(V) \tag{5}$$

For the loss function $R_0$, we use the squared loss $R_0(f(x_i), y_i) = (f(x_i) - y_i)^2$ for simplicity.

## 3 Implementation

Since the manifold $\mathcal{M}$ is unknown, the function $f$ which minimizes (5) can not be directly solved. In this section, we discuss how to discretize the continuous objective function (5).

### 3.1 Vector Field Representation

Given $l$ labeled data points $(x_i, y_i)_{i=1}^l$ and $n - l$ unlabeled points $x_{l+1}, \dots, x_n$ in $\mathbb{R}^m$. Let $f_i = f(x_i), i = 1, \dots, n$, our goal is to learn a function $f = (f_1, \dots, f_n)^T$. We first construct a nearest neighbor graph by either $\epsilon$-neighborhood or $k$ nearest neighbors. Let $x_i \sim x_j$ denote that $x_i$ and

$x_j$ are neighbors. For each point $x_i$, we estimate its tangent space $T_{x_i}\mathcal{M}$ by performing PCA on its neighborhood. We choose the largest $d$ eigenvectors as the bases since $T_{x_i}\mathcal{M}$ is $d$ dimensional. Let $T_i \in \mathbb{R}^{m \times d}$ be the matrix whose columns constitute an orthonormal basis for $T_{x_i}\mathcal{M}$. It is easy to show that $P_i = T_i T_i^T$ is the *unique* orthogonal projection from $\mathbb{R}^m$ onto the tangent space $T_{x_i}\mathcal{M}$ [11]. That is, for any vector $a \in \mathbb{R}^m$, we have $P_i a \in T_{x_i}\mathcal{M}$ and $(a - P_i a) \perp P_i a$.

Let $V$ be a vector field on the manifold. For each point $x_i$, let $V_{x_i}$ denote the value of the vector field $V$ at $x_i$, and $\nabla V|_{x_i}$ denote the value of $\nabla V$ at $x_i$. According to the definition of vector field, $V_{x_i}$ should be a vector in tangent space $T_{x_i}\mathcal{M}$. Therefore, it can be represented by the local coordinates of the tangent space, $V_{x_i} = T_i v_i$, where $v_i \in \mathbb{R}^d$. We define $\mathbb{V} = \left(v_1^T, \ldots, v_n^T\right)^T \in \mathbb{R}^{dn}$. That is, $\mathbb{V}$ is a $dn$-dimensional big column vector which concatenates all the $v_i$'s. In the following, we first discretize our objective function $E(f, V)$, and then minimize it to obtain $f$ and $\mathbb{V}$.

## 3.2 Gradient Field Computation

In order to discretize $R_1(f, V)$, we first discuss the Taylor expansion of $f$ on the manifold. Let $\exp_x$ denote the exponential map at $x$. The exponential map $\exp_x : T_x\mathcal{M} \to \mathcal{M}$ maps the tangent space $T_x\mathcal{M}$ to the manifold $\mathcal{M}$. Let $a \in T_x\mathcal{M}$ be a tangent vector. Then there is a *unique* geodesic $\gamma_a$ satisfying $\gamma_a(0) = x$ with the initial tangent vector $\gamma_a'(0) = a$. The corresponding exponential map is defined as $\exp_x(ta) = \gamma_a(t)$, $t \in [0, 1]$. Locally, the exponential map is a diffeomorphism.

Note that $f \circ \exp_x : T_x\mathcal{M} \to \mathbb{R}$ is a smooth function on $T_x\mathcal{M}$. Then the following Taylor expansion of $f$ holds:
$$f(\exp_x(a)) \approx f(x) + \langle \nabla f(x), a \rangle, \tag{6}$$
where $a \in T_x\mathcal{M}$ is a sufficiently small tangent vector. In the discrete case, let $\exp_{x_i}$ denote the exponential map at $x_i$. Since $\exp_{x_i}$ is a diffeomorphism, there exists a tangent vector $a_{ij} \in T_{x_i}\mathcal{M}$ such that $\exp_{x_i}(a_{ij}) = x_j$. According to the definition of exponential map, $\|a_{ij}\|$ equals to the geodesic distance between $x_i$ and $x_j$, which can be denoted as $d_{ij}$. Let $e_{ij}$ be the unit vector in the direction of $a_{ij}$, i.e., $e_{ij} = a_{ij}/d_{ij}$. We approximate $a_{ij}$ by projecting the vector $x_j - x_i$ to the tangent space, i.e., $a_{ij} = P_i(x_j - x_i)$. Therefore, Eq. (6) can be rewritten as follows:
$$f(x_j) = f(x_i) + \langle \nabla f(x_i), P_i(x_j - x_i) \rangle \tag{7}$$

Since $f$ is unknown, $\nabla f$ is also unknown. In the following, we discuss how to compute $\|\nabla f(x_i) - V_{x_i}\|^2$ discretely. We first show that the vector norm can be computed by an integral on a unit sphere, where the unit sphere can be discretely approximated by a neighborhood.

Let $u$ be a unit vector on tangent space $T_x\mathcal{M}$, then we have (see the exercise 1.12 in [12])
$$\frac{1}{\omega_d} \int_{S^{d-1}} \langle X, u \rangle^2 d\delta(X) = 1 \tag{8}$$
where $S^{d-1}$ is the unit $(d-1)$-sphere, $d\omega_d$ is its volume, and $d\delta$ is its volume form. Let $\partial_i$, $i = 1, \ldots, d$, be an orthonormal basis of $T_x\mathcal{M}$. Then for any vector $b \in T_x\mathcal{M}$, it can be written as $b = \sum_{i=1}^d b^i \partial_i$. Furthermore, we have
$$\|b\|^2 = \sum_{i=1}^d (b^i)^2 = \sum_{i=1}^d (b^i)^2 \frac{1}{\omega_d} \int_{S^{d-1}} \langle X, \partial_i \rangle^2 d\delta(X) = \frac{1}{\omega_d} \int_{S^{d-1}} \langle X, b \rangle^2 d\delta(X)$$
From Eq. (7), we can see that
$$\langle \nabla f(x_i), P_i(x_j - x_i) \rangle = f(x_j) - f(x_i).$$
Thus, we have
$$\begin{aligned}
\|\nabla f(x_i) - V_{x_i}\|^2 &= \frac{1}{\omega_d} \int_{S^{d-1}} \langle X, \nabla f(x_i) - V_{x_i} \rangle^2 d\delta(X) \\
&\approx \sum_{j \sim i} \langle e_{ij}, \nabla f(x_i) - V_{x_i} \rangle^2 = \sum_{j \sim i} w_{ij} \langle a_{ij}, \nabla f(x_i) - V_{x_i} \rangle^2 \\
&= \sum_{j \sim i} w_{ij} \langle P_i(x_j - x_i), \nabla f(x_i) - V_{x_i} \rangle^2 \\
&= \sum_{j \sim i} w_{ij} \left( (P_i(x_j - x_i))^T V_{x_i} - f(x_j) + f(x_i) \right)^2.
\end{aligned} \tag{9}$$

where $w_{ij} = d_{ij}^{-2}$. The weight $w_{ij}$ can be approximated either by heat kernel weight $\exp(-\|x_i - x_j\|^2/\delta)$ or simply by $0 - 1$ weight. Then $R_1$ reduces to the following:

$$R_1(f, \mathbb{V}) = \sum_i \sum_{j \sim i} w_{ij} \left( (x_j - x_i)^T T_i v_i - f_j + f_i \right)^2 \tag{10}$$

## 3.3 Parallel Field Computation

As discussed before, we hope the vector field to be as parallel as possible on the manifold. In the discrete case, $R_2$ becomes

$$R_2(\mathbb{V}) = \sum_{i=1}^{n} \|\nabla V|_{x_i}\|_F^2 \tag{11}$$

In the following, we discuss how to approximate $\|\nabla V|_{x_i}\|_F^2$ for a given point $x_i$. Since we do not know $\nabla_{\partial_i} V$ for a given basis $\partial_i$, $\|\nabla V|_{x_i}\|_F^2$ cannot be computed according to Eq. (4). We define a $(0, 2)$ symmetric tensor $\alpha$ as $\alpha(X, Y) = g(\nabla_X V, \nabla_Y V)$, where $X$ and $Y$ are vector fields on the manifold. We have $\text{Trace}(\alpha) = \sum_{i=1}^{d} g(\nabla_{\partial_i} V, \nabla_{\partial_i} V) = \|\nabla V\|_F^2$, where $\partial_1, \ldots, \partial_d$ is an orthonormal basis of the tangent space. For the trace of $\alpha$, we have the following geometric interpretation (see the exercise 1.12 in [12]):

$$\text{Trace}(\alpha) = \frac{1}{\omega_d} \int_{S^{d-1}} \alpha(X, X) d\delta(X) \tag{12}$$

where $S^{d-1}$ is the unit $(d-1)$-sphere, $d\omega_d$ its volume, and $d\delta$ its volume form. So for a given point $x_i$, we can approximate $\|\nabla V|_{x_i}\|$ by the following

$$\|\nabla V|_{x_i}\|_F^2 = \text{Trace}(\alpha)_{x_i} = \frac{1}{\omega_d} \int_{S^{d-1}} \alpha(X, X)|_{x_i} d\delta(X) \approx \sum_{j \sim i} \alpha(e_{ij}, e_{ij}) = \sum_{j \sim i} \|\nabla_{e_{ij}} V\|^2 \tag{13}$$

Then we discuss how to discretize $\nabla_{e_{ij}} V$. Given $e_{ij} \in T_{x_i} \mathcal{M}$, there exists a unique geodesic $\gamma(t)$ which satisfies $\gamma(0) = x_i$ and $\gamma'(0) = e_{ij}$. Then the covariant derivative of vector field $V$ along $e_{ij}$ is given by (please see Fig. 1)

$$\nabla_{e_{ij}} V = P_i \left( \frac{dV}{dt}|_{t=0} \right) = P_i \lim_{t \to 0} \frac{V(\gamma(t)) - V(\gamma(0))}{t} \approx P_i \frac{(V_{x_j} - V_{x_i})}{d_{ij}} = \sqrt{w_{ij}} (P_i V_{x_j} - V_{x_i})$$

Combining Eq. (13), $R_2$ becomes:

$$R_2(\mathbb{V}) = \sum_i \sum_{j \sim i} w_{ij} \|P_i T_j v_j - T_i v_i\|^2 \tag{14}$$

## 3.4 Objective Function in the Discrete Form

Let $\mathbb{I}$ denote a $n \times n$ diagonal matrix where $\mathbb{I}_{ii} = 1$ if $x_i$ is labeled and $\mathbb{I}_{ii} = 0$ otherwise. And let $y \in \mathbb{R}^n$ be a column vector whose $i$-th element is $y_i$ if $x_i$ is labeled and $0$ otherwise. Then

$$R_0(f) = \frac{1}{l} (f - y)^T \mathbb{I} (f - y) \tag{15}$$

Combining $R_1$ in Eq. (10) and $R_2$ in Eq. (14), the final objective function in the discrete form can be written as follows:

$$E(f, \mathbb{V}) = \frac{1}{l} (f - y)^T \mathbb{I} (f - y) + \lambda_1 \sum_i \sum_{j \sim i} w_{ij} \left( (x_j - x_i)^T T_i v_i - f_j + f_i \right)^2$$
$$+ \lambda_2 \sum_i \sum_{j \sim i} w_{ij} \|P_i T_j v_j - T_i v_i\|^2 \tag{16}$$

## 3.5 Optimization

In this subsection, we discuss how to solve the optimization problem (16).

Let $L$ denote the Laplacian matrix of the graph with weights $w_{ij}$. Then we can rewrite $R_1$ as follows:

$$R_1(f, \mathbb{V}) = 2f^T L f + \sum_i \sum_{j \sim i} w_{ij} \left( (x_j - x_i)^T T_i v_i \right)^2 - 2 \sum_i \sum_{j \sim i} w_{ij} (x_j - x_i)^T T_i v_i s_{ij}^T f$$

where $s_{ij} \in \mathbb{R}^n$ is a selection vector of all zero elements except for the $i$-th element being $-1$ and the $j$-th element being 1. Then the partial derivative of $R_1$ with respect to the variable $v_i$ is

$$\frac{\partial R_1(f, \mathbb{V})}{\partial v_i} = 2 \sum_{j \sim i} w_{ij} T_i^T (x_j - x_i)(x_j - x_i)^T T_i v_i - 2 \sum_{j \sim i} w_{ij} T_i^T (x_j - x_i) s_{ij}^T f$$

Thus we get

$$\frac{\partial R_1(f, \mathbb{V})}{\partial \mathbb{V}} = 2G\mathbb{V} - 2Cf \tag{17}$$

where $G$ is a $dn \times dn$ block diagonal matrix, and $C = [C_1^T, \ldots, C_n^T]^T$ is a $dn \times n$ block matrix. Denote the $i$-th $d \times d$ diagonal block of $G$ by $G_{ii}$ and the $i$-th $d \times n$ block of $C$ by $C_i$, we have

$$G_{ii} = \sum_{j \sim i} w_{ij} T_i^T (x_j - x_i)(x_j - x_i)^T T_i \tag{18}$$

$$C_i = \sum_{j \sim i} w_{ij} T_i^T (x_j - x_i) s_{ij}^T \tag{19}$$

The partial derivative of $R_1$ with respect to the variable $f$ is

$$\frac{\partial R_1(f, \mathbb{V})}{\partial f} = 4Lf - 2C^T \mathbb{V} \tag{20}$$

Similarly, we can compute the partial derivative of $R_2$ with respect to the variable $v_i$:

$$\frac{\partial R_2(\mathbb{V})}{\partial v_i} = 2 \sum_{j \sim i} w_{ij} \left( (T_i^T T_j T_j^T T_i + I) v_i - 2 T_i^T T_j v_j \right) = 2 \sum_{j \sim i} w_{ij} \left( (Q_{ij} Q_{ij}^T + I) v_i - 2 Q_{ij} v_j \right)$$

where $Q_{ij} = T_i^T T_j$. Thus we obtain

$$\frac{\partial R_2}{\partial \mathbb{V}} = 2B\mathbb{V} \tag{21}$$

where $B$ is a $dn \times dn$ sparse block matrix. If we index each $d \times d$ block by $B_{ij}$, then for $i, j = 1, \ldots, n$, we have

$$B_{ii} = \sum_{j \sim i} w_{ij} (Q_{ij} Q_{ij}^T + I) \tag{22}$$

$$B_{ij} = \begin{cases} -2w_{ij} Q_{ij}, & \text{if } x_i \sim x_j \\ 0, & \text{otherwise} \end{cases} \tag{23}$$

Notice that $\frac{\partial R_0}{\partial f} = 2\frac{1}{l}\mathbb{I}(f - y)$. Combining Eq. (17), Eq. (20) and Eq. (21), we have

$$\frac{\partial E(f, \mathbb{V})}{\partial f} = \frac{\partial R_0}{\partial f} + \lambda_1 \frac{\partial R_1}{\partial f} + \lambda_2 \frac{\partial R_2}{\partial f} = 2(\frac{1}{l}\mathbb{I} + 2\lambda_1 L)f - 2\lambda_1 C^T \mathbb{V} - 2\frac{1}{l} y \tag{24}$$

$$\frac{\partial E(f, \mathbb{V})}{\partial \mathbb{V}} = \frac{\partial R_0}{\partial \mathbb{V}} + \lambda_1 \frac{\partial R_1}{\partial \mathbb{V}} + \lambda_2 \frac{\partial R_2}{\partial \mathbb{V}} = -2\lambda_1 Cf + 2(\lambda_1 G + \lambda_2 B)\mathbb{V} \tag{25}$$

Requiring that the derivatives vanish, we finally get the following linear system

$$\begin{pmatrix} \frac{1}{l}\mathbb{I} + 2\lambda_1 L & -\lambda_1 C^T \\ -\lambda_1 C & \lambda_1 G + \lambda_2 B \end{pmatrix} \begin{pmatrix} f \\ \mathbb{V} \end{pmatrix} = \begin{pmatrix} \frac{1}{l} y \\ 0 \end{pmatrix} \tag{26}$$

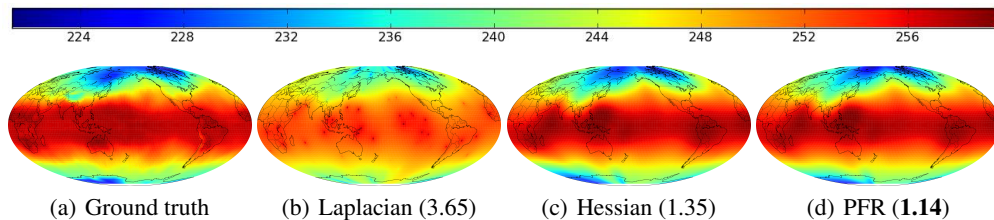

|     (a) Ground truth     |     (b) Laplacian (3.65)     |     (c) Hessian (1.35)     |     (d) PFR (**1.14**)     |

Figure 2: Global temperature prediction. Regression on the satellite measurement of temperatures in the middle troposphere. 1% samples are randomly selected as training data. The ground truth is shown in (a). The colors indicate temperature values (in Kelvin). The regression results are visualized in (b)∼(d). The numbers in the captions are the mean absolute prediction errors.

## 4   Related Work and Discussion

The approximation of the Laplacian operator using the graph Laplacian [5] has enjoyed a great success in the last decade. Some theoretical results [13, 14] also show the consistency of the approximation. One of the most important features of the graph Laplacian is that it is coordinate free. That is, it does not depend on any special coordinate system.

The estimation of Hessian is very difficult and there is few work on it. Previous approaches [2, 15] first estimate normal coordinates in the tangent space, and then estimate the first order derivative at each point, which is a matrix pseudo-inversion problem. One major limitation of this is that when the number of nearest neighbors $k$ is larger than $d + \frac{d(d+1)}{2}$, where $d$ is the dimension of the manifold, the estimation will be inaccurate and unstable [15]. This is contradictory to the asymptotic case, since it is not desirable that $k$ is bounded by a finite number when the data is sufficiently dense. In contrast, our method is coordinate free. Also, we directly estimate the norm of the second order derivative instead of trying to estimate its coefficients, which turns out to be an integral problem over the neighboring points. We only need to do simple matrix multiplications to approximate the integral at each point, but do not have to solve matrix inversion problems. Therefore, asymptotically, we would expect our method to be much more accurate and robust for the approximation of the norm of the second order derivative.

## 5   Experiments

In this section, we compare our proposed Parallel Field Regularization (PFR) algorithm with two state-of-the-art semi-supervised regression methods: Laplacian regularized transduction (Laplacian) [8] and Hessian regularized transduction (Hessian)[1] [15], respectively. Our experiments are carried out on two real-world data sets. Regularization parameters for all algorithms are chosen via cross-validation.

### 5.1   Global Temperature

In this test, we perform regression on the earth surface, which is a 2D sphere manifold. We try to predict the satellite measurement of temperatures in the middle troposphere in Dec. 2004[2], which contains 9504 valid temperature measurements. The coordinates (latitude, longitude) of the measurements are used as features and the corresponding temperature values are the responses. The dimension of manifold is set to 2 and the number of nearest neighbors is set to 6 in graph construction. We randomly select 1% of the samples as labeled data, and compare the predicted temperature values with the ground truth on the rest of the data.

The regression results are shown in Fig. 2. The numbers in the captions indicate the mean absolute prediction errors generated by different algorithms. It can be seen from the visualization result that

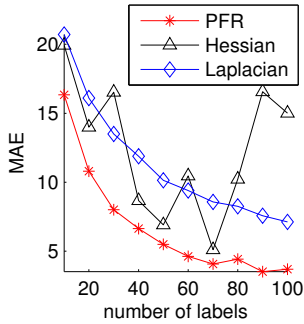

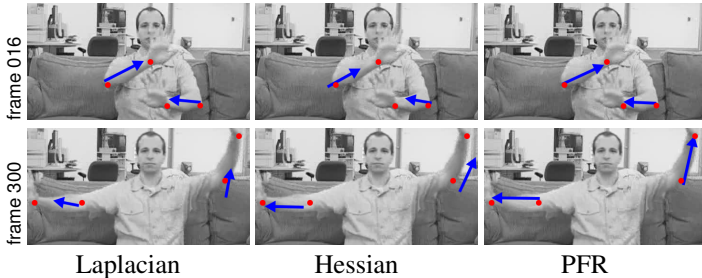

Figure 4: The examples of regression results on the moving hand data set. 60 labeled samples are used for training. Each row shows the results obtained via the three algorithms for a frame. In each image, the red dots indicate the ground truth positions we labeled manually, and the blue arrows show the positions predicted by different algorithms.

Figure 3: Results on the moving hand dataset.

Hessian and PFR perform better than Laplacian. Furthermore, from the prediction error, we can see that PFR outperforms Hessian.

## 5.2 Positions of Moving Hands

In this subsection, we perform experiments using a video of a subject sitting in a sofa and waving his arms [1]. Our goal is to predict the positions of the (left and right) elbows and wrists. We extract the first 500 frames of the video and manually label the positions of the elbows and wrists. We scale each frame to size of $120 \times 90$ and use the raw pixels (10800-dimensional vectors) as the features. The response for each frame is a 8-dimensional vector whose elements are the 2D coordinates of the elbows and wrists. Since there are 8 free parameters, we set the dimension of manifold to 8. We use 18 nearest neighbors in graph construction.

We run the experiments with different numbers of labeled frames. For each given number of labeled frames, we perform 10 tests with randomly selected labeled set. The average of the mean absolute error (MAE) for each test is calculated. The final result is shown in Fig. 3. As can be seen, PFR consistently outperforms the other two algorithms. Laplacian yields high MAE. Hessian is very unstable on this dataset, and the results vary drastically with different numbers of labels.

We also show some example frames in Fig. 4. The red dots in the figures indicate the ground truth positions and the blue arrows are drawn by connecting the positions of elbows and wrists predicted by different algorithms. Again we can verify that PFR performs better than the other two algorithms.

## 6 Conclusion

In this paper, we propose a novel semi-supervised learning algorithm from the vector field perspective. We show the relationship between vector fields and functions on the manifold. The parallelism of the vector field is used to measure the linearity of the target prediction function. Parallel fields are one kind of special vector fields on the manifold, which have very nice properties. It is interesting to explore other kinds of vector fields to facilitate learning on manifolds. Moreover, vector fields can also be used to study the geometry and topology of the manifold. For example, Poincaré-Hopf theorem tells us that the sum of the indices over all the isolated zeroes of a vector field equals to the Euler characteristic of the manifold, which is a very important topological invariant.

## Acknowledgments

This work was supported by the National Natural Science Foundation of China under Grant 61125203, the National Basic Research Program of China (973 Program) under Grant 2012CB316404, and the National Natural Science Foundation of China under Grants 90920303 and 60875044.

## Footnotes

[1]We use the code from the authors downloadable from `http://www.ml.uni-saarland.de/code/HessianSSR/HessianSSR.html`.

[2]`http://www.remss.com/msu/`.

[1]The video is obtained from `http://www.csail.mit.edu/~rahimi/manif`.

# References

[1] J. Tenenbaum, V. de Silva, and J. Langford. A global geometric framework for nonlinear dimensionality reduction. *Science*, 290(5500):2319–2323, 2000.

[2] D. L. Donoho and C. E. Grimes. Hessian eigenmaps: Locally linear embedding techniques for high-dimensional data. *Proceedings of the National Academy of Sciences of the United States of America*, 100(10):5591–5596, 2003.

[3] S. Roweis and L. Saul. Nonlinear dimensionality reduction by locally linear embedding. *Science*, 290(5500):2323–2326, 2000.

[4] M. Belkin and P. Niyogi. Laplacian eigenmaps and spectral techniques for embedding and clustering. In *Advances in Neural Information Processing Systems 14*, pages 585–591. 2001.

[5] Fan R. K. Chung. *Spectral Graph Theory*, volume 92 of *Regional Conference Series in Mathematics*. AMS, 1997.

[6] X. Zhu and J. Lafferty. Semi-supervised learning using gaussian fields and harmonic functions. In *Proc. of the 20th Internation Conference on Machine Learning*, 2003.

[7] D. Zhou, O. Bousquet, T.N. Lal, J. Weston, and B. Schölkopf. Learning with local and global consistency. In *Advances in Neural Information Processing Systems 16*, 2003.

[8] Mikhail Belkin, Irina Matveeva, and Partha Niyogi. Regularization and semi-supervised learning on large graphs. In *Conference on Learning Theory*, pages 624–638, 2004.

[9] John Lafferty and Larry Wasserman. Statistical analysis of semi-supervised regression. In *Advances in Neural Information Processing Systems 20*, pages 801–808, 2007.

[10] P. Petersen. *Riemannian Geometry*. Springer, New York, 1998.

[11] G. H. Golub and C. F. Van Loan. *Matrix computations*. Johns Hopkins University Press, 3rd edition, 1996.

[12] B. Chow, P. Lu, and L. Ni. *Hamilton's Ricci Flow*. AMS, Providence, Rhode Island, 2006.

[13] Mikhail Belkin and Partha Niyogi. Towards a theoretical foundation for laplacian-based manifold methods. In *Conference on Learning Theory*, pages 486–500, 2005.

[14] Matthias Hein, Jean yves Audibert, and Ulrike Von Luxburg. From graphs to manifolds - weak and strong pointwise consistency of graph laplacians. In *Conference on Learning Theory*, pages 470–485, 2005.

[15] K. I. Kim, F. Steinke, and M. Hein. Semi-supervised regression using hessian energy with an application to semi-supervised dimensionality reduction. In *Advances in Neural Information Processing Systems 22*, pages 979–987. 2009.

